# Entropy Estimations Using Correlated Symmetric Stable Random Projections

**Ping Li**
Department of Statistical Science
Cornell University
Ithaca, NY 14853
pingli@cornell.edu

**Cun-Hui Zhang**
Department of Statistics and Biostatistics
Rutgers University
New Brunswick, NJ 08901
czhang@stat.rutgers.edu

## Abstract

Methods for efficiently estimating Shannon entropy of data streams have important applications in learning, data mining, and network anomaly detections (e.g., the DDoS attacks). For nonnegative data streams, the method of *Compressed Counting (CC)* [11, 13] based on maximally-skewed stable random projections can provide accurate estimates of the Shannon entropy using small storage. However, CC is no longer applicable when entries of data streams can be below zero, which is a common scenario when comparing two streams. In this paper, we propose an algorithm for entropy estimation in general data streams which allow negative entries. In our method, the Shannon entropy is approximated by the finite difference of two **correlated** frequency moments estimated from correlated samples of **symmetric** stable random variables. Interestingly, the estimator for the moment we recommend for entropy estimation barely has bounded variance itself, whereas the common geometric mean estimator (which has bounded higher-order moments) is not sufficient for entropy estimation. Our experiments confirm that this method is able to well approximate the Shannon entropy using small storage.

## 1  Introduction

Computing the Shannon entropy in massive data have important applications in neural computation [17], graph estimation [5], query logs analysis in Web search [14], network anomaly detection [21], etc. (See NIPS2003 workshop on entropy estimation www.menem.com/~ilya/pages/NIPS03). In modern applications, as massive datasets are often generated in a streaming fashion, entropy estimation in data streams has become a challenging and interesting problem.

### 1.1  Data Streams

Massive data generated in a streaming fashion are difficult to transmit and store [15], as the processing is often done **on the fly** in **one-pass** of the data. The problem of "scaling up for high dimensional data and high speed data streams" is among the "ten challenging problems in data mining research" [20]. Mining data streams at petabyte scale has become an important research area [1], as network data can easily reach that scale [20].

In the standard **turnstile model** [15], a data stream is a vector $A_t$ of length $D$, where $D = 2^{64}$ or even $D = 2^{128}$ is possible in network applications, e.g., (a pair of) IP addresses + port numbers. At time $t$, there is an input stream $a_t = (i_t, I_t)$, $i_t \in [1,\ D]$ which updates $A_t$ by a linear rule:

$$A_t[i_t] = A_{t-1}[i_t] + I_t. \tag{1}$$

where $I_t$ is the increment/decrement of package size at $t$. For network traffic, normally $A_t[i] \geq 0$, which is called the **strict turnstile model** and suffices for describing certain natural phenomena. On the other hand, the general turnstile model (which allows $A_t[i] < 0$) is often used for comparing two streams, e.g., in network OD (origin-destination) flow analysis [21].

An important task is to compute the $\alpha$-th frequency moment $F_{(\alpha)}$ and the Shannon entropy $H$:

$$F_{(\alpha)} = \sum_{i=1}^{D} |A_t[i]|^{\alpha}, \qquad H = -\sum_{i=1}^{D} \frac{|A_t[i]|}{F_1} \log \frac{|A_t[i]|}{F_1}, \tag{2}$$

The exact computation of these summary statistics is not feasible because to do so one has to store the entire vector $A_t$ of length $D$, as the entries are time-varying. Also, many applications (such as anomaly detections of network traffic) require computing the summary statistics in real-time.

## 1.2 Network Measurement, Monitoring, and Anomaly Detection

Network traffic is a typical example of high-rate data streams. Industries are now prepared to move to 100 Gbits/second or Terabit/second Ethernet. An effective and reliable measurement of network traffic in real-time is crucial for anomaly detection and network diagnosis; and one such measurement metric is the Shannon entropy [4, 8, 19, 2, 9, 21]. The exact entropy measurement in real-time on high-speed links is however computationally prohibitive.

The *Distributed Denial of Service (DDoS)* attack is a representative example of network anomalies. A DDoS attack attempts to make computers unavailable to intended users, either by forcing users to reset the computers or by exhausting the resources of service-hosting sites. For example, hackers may maliciously saturate the victim machines by sending many external communication requests. DDoS attacks typically target sites such as banks, credit card payment gateways, or military sites. A DDoS attack normally changes the statistical distribution of network traffic, which could be reliably captured by the abnormal variations in the measurements of Shannon entropy [4]. See Figure 1 for an illustration.

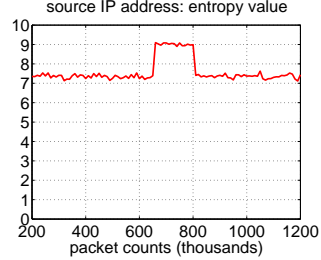

Figure 1: This plot is reproduced from a DARPA conference [4]. One can view x-axis as the surrogate for time. Y-axis is the measured Shannon entropy, which exhibited a sudden sharp change at the time when an attack occurred.

Apparently, the entropy measurements do not have to be "perfect" for detecting attacks. It is however crucial that the algorithms should be computationally efficient (i.e., real-time and one-pass) at low memory cost, because the traffic data generating by large high-speed networks are enormous and transient.

## 1.3 Symmetric Stable Random Projections and Entropy Estimation Using Moments

It turns out that, for $0 < \alpha \leq 2$, one can use stable random projections to compute $F_{(\alpha)}$ efficiently because the Turnstile model (1) is a linear model and the random projection operation is also linear (i.e., vector-matrix multiplication) [7]. Conceptually, we multiply the data stream vector $A_t \in \mathbb{R}^D$ by a random matrix $\mathbf{R} \in \mathbb{R}^{D \times k}$, resulting in a vector $X = A_t \times \mathbf{R} \in \mathbb{R}^k$ with entries

$$x_j = [A_t \times \mathbf{R}]_j = \sum_{i=1}^{D} r_{ij} A_t[i], \ \ j = 1, 2, ..., k$$

where $r_{ij} \sim S(\alpha, 1)$ is a symmetric $\alpha$-stable random variable with unit scale [3, 22]: $E(e^{r_{ij}t}) = e^{-|t|^\alpha}$. The standard normal (or Cauchy) distribution is a special case with $\alpha = 2$ (or $\alpha = 1$).

In data stream computations, the matrix $\mathbf{R}$ is not materialized. The standard procedure is to (re)generate entries of $\mathbf{R}$ on-demand [7] using pseudo-random numbers [16]. Thus, we only need to store $X \in \mathbb{R}^k$. When a stream element $a_t = (i_t, I_t)$ arrives, one updates the entries of $X$:

$$x_j \leftarrow x_j + I_t r_{i_t j}, \quad j = 1, 2, ..., k. \tag{3}$$

By property of stable distributions, the samples $x_j$, $j = 1$ to $k$, are also i.i.d. stable

$$x_j = \sum_{i=1}^{D} r_{ij} A_t[i] \sim S\left(\alpha, F_{(\alpha)} = \sum_{i=1}^{D} |A_t[i]|^\alpha\right) \tag{4}$$

Therefore, the task boils down to estimating the scale parameter from $k$ i.i.d. stable samples.

Because the Shannon entropy is essentially the derivative of the frequency moment at $\alpha = 1$, the popular approach is to approximate the Shannon entropy by the Tsallis entropy [18]:

$$T_\alpha = \frac{1}{\alpha - 1} \left(1 - \frac{F_{(\alpha)}}{F_{(1)}^\alpha}\right). \tag{5}$$

which approaches the Shannon entropy $H$ as $\alpha \to 1$. [21] used a slight variant of (5) but the difference is not essential.[1] In their approach, $F_{(\alpha)}$ and $F_{(1)}$ are first estimated separately from

two independent sets of samples. The estimated moments are then plugged in (5) to estimate the Shannon entropy $H$. Immediately, we can see the problem here: the variance of the estimated $T_{(\alpha)}$ might be proportional to $\frac{1}{(\alpha-1)^2} = \frac{1}{\Delta^2}$. (Recall $var(cX) = c^2 var(X)$).

One question is how to choose $\alpha$ (i.e., $\Delta$). [6] proposed a conservative criterion by choosing $\alpha$ according to the worst case bias $|H - T_\alpha|$. One can verify that $\Delta = |1 - \alpha| < 10^{-7}$ is likely in [6]. In other words, the required sample size could be $O\left(10^{14}\right)$. In practice, [21] exploited the bias-variance tradeoff but they still had to use an excessive number of samples, e.g., $10^6$. In comparison, using our proposed approach, it appears that $100 \sim 1000$ samples might be sufficient.

## 1.4 Our Proposal

We have made **two key** contributions. Firstly, instead of estimating $F_{(\alpha)}$ and $F_{(1)}$ separately using two independent sets of samples, we make them **highly positively correlated**. Intuitively, if the two consistent estimators, denoted by $\hat{F}_{(\alpha)}$ and $\hat{F}_{(1)}$ respectively, are highly positively correlated, then possibly their ratio $\frac{\hat{F}_{(\alpha)}}{\hat{F}_{(1)}^\alpha}$ can be close to 1 with small variance. Ideally, if $Var\left(\frac{\hat{F}_{(\alpha)}}{\hat{F}_{(1)}^\alpha}\right) = O\left(\Delta^2\right)$, the variance of the estimated Tsallis entropy $\hat{T}_\alpha = \frac{1}{\alpha-1}\left(1 - \frac{\hat{F}_{(\alpha)}}{\hat{F}_{(1)}^\alpha}\right)$ will be essentially independent of $\Delta$.

It turns out that finding an estimator with $Var\left(\frac{\hat{F}_{(\alpha)}}{\hat{F}_{(1)}^\alpha}\right) = O\left(\Delta^2\right)$ was not straightforward. It is known that around $\alpha = 1$, the *geometric mean* estimator [10] is nearly statistically optimal. Interestingly, our analysis and simulation show that using the geometric mean estimator, we can essentially only achieve $Var\left(\frac{\hat{F}_{(\alpha)}}{\hat{F}_{(1)}^\alpha}\right) = O\left(\Delta\right)$, which, albeit a large improvement, is not small sufficient to cancel the $O\left(\frac{1}{\Delta^2}\right)$ term. Therefore, our second key component is a new estimator of $T_\alpha$ using a moment estimator which does not have (or barely has) finite variance. Even though such an estimator is not good for estimating the single moment compared to the geometric mean, due to the high correlation, the ratio $\frac{\hat{F}_{(\alpha)}}{\hat{F}_{(1)}^\alpha}$ is still very well-behaved and its variance is essentially $O\left(\Delta^2\right)$, as shown in our theoretical analysis and experiments.

## 1.5 Compressed Counting (CC) for Nonnegative Data Streams

The recent work [13] on *Compressed Counting (CC)* [11] provides an ideal solution to the problem of entropy estimation in **nonnegative** data streams. Basically, for nonnegative data streams, i.e., $A_t[i] \geq 0$ at all times and all locations, we can compute the first moment easily, because

$$F_{(1)} = \sum_{i=1}^{D} |A_t[i]| = \sum_{i=1}^{D} A_t[i] = \sum_{s=0}^{t} I_s \tag{6}$$

where $I_s$ is the increment/decrement at time $s$. In other words, we just need a single counter to accumulate all the increments $I_s$. This observation lead to the conjecture that estimating $F_{(\alpha)}$ should be also easy if $\alpha \approx 1$, which consequently lead to the development of Compressed Counting which used maximally-skewed stable random projections instead of symmetric stable projections. The most recent work of CC [13] provided a new moment estimator to achieve the variance $\propto O\left(\Delta^2\right)$.

Unfortunately, for general data streams where entries can be negative, we have to resort to symmetric stable random projections. Fundamentally, the reason that skewed projections work well on nonnegative data streams is because the data themselves are skewed. However, when we compare two streams, the data become more or less symmetric and hence we must use symmetric projections.

## 1.6 Why Comparing the Difference of Two Streams?

In machine learning research and practice, people routinely use the difference between feature vectors. [21] used the difference between data streams from a slightly different motivation.

The goal of [21] is to measure the entropies of all OD pairs (origin-destination) in a network, because entropy measurements are crucial for detecting anomaly events such as DDoS attacks and network failures. They argued that the change of entropy of the traffic distribution may be invisible (i.e., too small to be detected) in the traditional volume matrix even during the time when an attack occurs. Instead, they proposed to measure the entropy from a number of locations across the network, i.e.,

by examining the entropy of every OD flow in the network. In a similar argument, a DDoS attack may be invisible in terms of the traffic volume change, if the attack is launched outside the network.

While [21] successfully demonstrated that measuring the Shannon entropy of OD flows is effective for detecting anomaly events, at that time they did not have the tools for efficiently estimating the entropy. Using symmetric stable random projections and independent samples, they needed a large number of samples (e.g., $10^6$) because their variance blows up at the rate of $O\left(\frac{1}{\Delta^2}\right)$.

For anomaly detection, reducing the sample size ($k$) is crucial because $k$ determines the storage and estimation speed; and it is often required to detect the events at real time. In addition, the pseudo-random numbers have to be (re)-generated on the fly, at a cost proportional to $k$.

## 2   Our Proposed Algorithm

Recall that a data stream is a long vector $A_t[i]$, $i = 1$ to $D$. At time $t$, an incoming element $a_t = (i_t, I_t)$ updates one entry: $A_t[i_t] \leftarrow A_{t-1}[i_t] + I_t$. Conceptually, we generate a random matrix $\mathbf{R} \in \mathbb{R}^{D \times k}$ whose entries are sampled from a stable distribution and multiply it with $A_t$: $X = A_t \times \mathbf{R}$. The matrix multiplication is linear and can be conducted incrementally as the new stream elements arrive. $\mathbf{R}$ is not materialized; its entries are re-generated on demand using pseudo-random numbers, as the standard practice in data stream computations [7]. Our method does not require $A_t[i] \geq 0$ and hence it can handle the difference between two streams (e.g., the OD flows).

### 2.1   The Symmetric Stable Law

Our work utilizes the symmetric stable distribution. We adopt the standard approach [3] to sample from the stable law $S(\alpha, 1)$ with index $\alpha$ and unit scale. We generate two independent random variables: $w \sim exp(1)$ and $u \sim unifom(-\pi/2, \pi/2)$ and feed them to a nonlinear transformation:

$$Z(\alpha) = g(w, u, \alpha) = \frac{\sin(\alpha u)}{(\cos u)^{1/\alpha}} \left[\frac{\cos(u - \alpha u)}{w}\right]^{(1-\alpha)/\alpha} \sim S(\alpha, 1), \qquad (7)$$

to obtain a sample from $S(\alpha, 1)$. An important property is that, for $-1 < \gamma < \alpha$, the moment exists: $E|Z|^\gamma = (2/\pi)\Gamma(1 - \gamma/\alpha)\Gamma(\gamma)\sin(\gamma\pi/2)$. For convenience, we define

$$G(\alpha, \gamma) = E|g(w, u, \alpha)|^\gamma = \frac{2}{\pi}\Gamma(1 - \gamma/\alpha)\Gamma(\gamma)\sin(\gamma\pi/2) \qquad (8)$$

### 2.2   Our Recommended Estimator

Conceptually, we have two matrices of i.i.d. random numbers:

$$w_{ij} \sim exp(1), \qquad u_{ij} \sim uniform(-\pi/2, \pi/2), \qquad i = 1, 2, ..., D, \quad j = 1, 2, ..., k, \qquad (9)$$

As new stream elements arrive, we incrementally maintain two sets of samples, i.e., for $i = 1$ to $k$,

$$x_j = \sum_{i=1}^{D} A_t[i]g(w_{ij}, u_{ij}, 1), \qquad y_j = \sum_{i=1}^{D} A_t[i]g(w_{ij}, u_{ij}, \alpha) \qquad (10)$$

Note that $x_j$ and $y_j$ are highly correlated because they are generated using the same random numbers (with different $\alpha$). However, $x_i$ and $y_j$ are independent if $i \neq j$.

**Our recommended estimator** of the Tsallis entropy $T_\alpha$ is

$$\hat{T}_{\alpha,0.5} = \frac{1}{\alpha - 1}\left(1 - \left(\frac{\sqrt{\pi}}{\Gamma\left(1 - \frac{1}{2\alpha}\right)}\frac{\sum_{j=1}^{k}\sqrt{|y_j|}}{\sum_{j=1}^{k}\sqrt{|x_j|}}\right)^{2\alpha}\right) \qquad (11)$$

where $\alpha = 1 + \Delta > 1$ and the meaning of 0.5 will soon be clear. When $\Delta$ is sufficiently small, the estimated Tsallis entropy will be sufficiently close to the Shannon entropy. A nice property is that its variance is free of $\frac{1}{\Delta}$ or $\frac{1}{\Delta^2}$ terms. While it is intuitively clear that it is beneficial to make $x_j$ and $y_j$ highly correlated for the sake of reducing the variance, it might not be as intuitive why $\hat{T}_{\alpha,0.5}$ (11) is a good estimator for the entropy. We will explain why the obvious geometric mean estimator [10] is not sufficient for entropy estimation.

## 3 The Geometric Mean Estimator

For estimating $F_{(\alpha)}$, the geometric mean estimator [10] is close to be statistically optimal (efficiency $\approx 80\%$) at $\alpha \approx 1$. Thus, it was our first attempt to test the following estimator of the Tsallis entropy:

$$\hat{T}_{\alpha,gm} = \frac{1}{\alpha-1}\left(1 - \frac{\hat{F}_{(\alpha),gm}}{\hat{F}^{\alpha}_{(1),gm}}\right), \quad \text{where} \quad \hat{F}_{(\alpha),gm} = \frac{\prod_{j=1}^{k}|y_j|^{\alpha/k}}{G^k(\alpha,\alpha/k)}, \quad \hat{F}_{(1),gm} = \frac{\prod_{j=1}^{k}|x_j|^{1/k}}{G^k(1,1/k)},$$

where $G()$ is defined in (8). After simplification, we obtain:

$$\hat{T}_{\alpha,gm} = \frac{1}{\alpha-1}\left(1 - \prod_{j=1}^{k}\left[\left|\frac{y_j}{x_j}\right|^{\alpha/k}\frac{G(1,1/k)}{G(\alpha,\alpha/k)}\right]\right). \tag{12}$$

### 3.1 Theoretical Analysis

The theoretical analysis of $\hat{T}_{(\alpha),gm}$, however, turns out to be difficult, as it requires computing

$$E\left[\left|\frac{y_j}{x_j}\right|^{s\alpha/k}\right] = E\left[\left|\frac{\sum_{i=1}^{D}A_t[i]g(w_{ij},u_{ij},\alpha)}{\sum_{i=1}^{D}A_t[i]g(w_{ij},u_{ij},1)}\right|^{s\alpha/k}\right], \quad s = 1, 2, \tag{13}$$

where $g()$ is defined in (7). We first provide the following Lemma:

**Lemma 1** *Let $w \sim exp(1)$ and $u \sim uniform(-\pi/2, \pi/2)$ be two independent variables. Let $\alpha = 1 + \Delta > 1$, for small $\Delta > 0$. Then, for $\gamma > -1$,*

$$E\left|\frac{g(w,u,\alpha)}{g(w,u,1)}\right|^{\gamma}$$
$$= 1 - 0.5772\gamma\Delta + 0.5772\gamma\Delta^2 - 1.6386\gamma\Delta^3 + 1.6822\gamma^2\Delta^2 + O\left(\gamma\Delta^4\right) + O\left(\gamma^2\Delta^3\right) \qquad \square \tag{14}$$

Note that we need to keep higher order terms in order to prove Lemma 2, to show the properties of the geometric mean estimator, when $D = 1$ (i.e., a stream with only one element).

**Lemma 2** *If $D = 1$, then*

$$E\left(\hat{T}_{\alpha,gm}\right) = \frac{1}{k}\frac{\pi^2}{2} - 2.0935\frac{\Delta}{k} + 1.0614\Delta^2 + O\left(\Delta^3\right) + O\left(\frac{\Delta^2}{k}\right) + O\left(\frac{1}{k^2}\right) \tag{15}$$

$$Var\left(\hat{T}_{\alpha,gm}\right) = \frac{3.3645}{k} + O\left(\frac{\Delta}{k}\right) + O\left(\frac{1}{k^2}\right) \qquad \square \tag{16}$$

When $D = 1$, we know $T_\alpha = H = 0$. In this case, the geometric mean estimator $\hat{T}_{\alpha,gm}$ is asymptotically unbiased with variance essentially free of $\frac{1}{\Delta}$, which is very encouraging.

Will this result in Lemma 2 extend to general $D$? The answer is no, even for $D = 2$, i.e.,

$$\frac{y_j}{x_j} = \frac{A_t[1]g(w_{1j},u_{1j},\alpha) + A_t[2]g(w_{2j},u_{2j},\alpha)}{A_t[1]g(w_{1j},u_{1j},1) + A_t[2]g(w_{2j},u_{2j},1)}$$

Because $g()$ is symmetric, it is possible that the denominator $A_t[1]g(w_{1j},u_{1j},1) + A_t[2]g(w_{2j},u_{2j},1)$ might be very small while the numerator $A_t[1]g(w_{1j},u_{1j},\alpha) + A_t[2]g(w_{2j},u_{2j},\alpha)$ is not too small. In other words, there will be more variations when $D > 1$. In fact, our experiments in Sec. 3.2 and the theoretical analysis of a more general estimator in Sec. 4 both reveal that the variance of $\hat{T}_{\alpha,gm}$ is essentially $O\left(\frac{1}{\Delta}\right)$, which is of course still a substantial improvement over the previous $O\left(\frac{1}{\Delta^2}\right)$ solution.

### 3.2 Experiments on the Geometric Mean Estimator (Correlated vs. Independent samples)

We present some experimental results for evaluating $\hat{T}_{\alpha,gm}$, to demonstrate that (i) using correlation does substantially reduce variance and hence reduces the required sample size, and (ii) the variance (or MSE, the mean square error) of $\hat{T}_{\alpha,gm}$ is roughly $O\left(\frac{1}{\Delta}\right)$.

We follow [13] by using static data to evaluate the accuracies of the estimators. The projected vector $X = A_t \times \mathbf{R}$ is the same at the end of the stream, regardless of whether it is computed at once (i.e., static) or incrementally (i.e., dynamic). Following [13], we selected 4 word vectors from a chunk of Web crawl data. For example, the entries for vector "REVIEW" are the numbers of occurrences of the word "REVIEW" in each document. We group these 4 vectors into 2 pairs: "THIS-HAVE" and "DO-REVIEW" and we estimate the Shannon entropies of the two resultant difference vectors.

Figure 2 presents the mean square errors (MSE) of the estimated Shannon entropy, i.e., $E(\hat{T}_\alpha - H)^2$, normalized by the truth ($H^2$). The left panels contain the results using independently sampling (i.e., the prior work [21]) and the geometric mean estimator. The middle panels contain the results using correlated sampling (i.e., this paper) and the geometric mean estimator (12). The right panels multiply the results of the middle panels by $\Delta$ to illustrate that the variance of the geometric mean estimator for entropy $\hat{T}_{\alpha,gm}$ is essentially $O\left(\frac{1}{\Delta}\right)$. See more experiments in Figure 3.

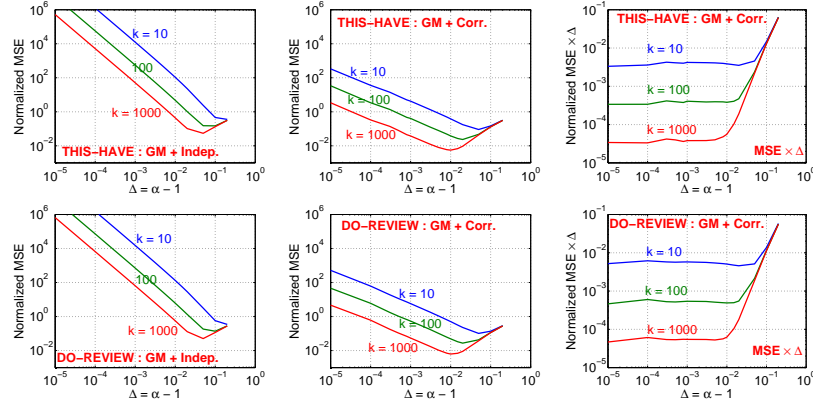

Figure 2: Two pairs of word vectors were selected. We conducted symmetric random projections using both independent sampling (left panels, as in [21]) and correlated sampling (middle panels, as our proposal). The Tsallis entropy (of the difference vector) is estimated using the geometric mean estimator (12) with three sample sizes $k = 10$, 100, and 1000. The normalized mean square errors (MSE: $E|\hat{T}_{\alpha,gm} - H|^2/H^2$) verify that correlated sampling reduces the errors substantially.

## 4   The General Estimator

Since the geometric mean estimator could not satisfactorily solve the entropy estimation problem, we resort to estimators which behave dramatically different from the geometric mean. Our recommended estimator $\hat{T}_{\alpha,0.5}$ as in (11) is a special case (for $\gamma = 0.5$) of a more general family of estimators [12], parameterized by $\gamma \in (0,\ 1)$:

$$\hat{T}_{\alpha,\gamma} = \frac{1}{\alpha-1}\left(1 - \frac{\hat{F}_{(\alpha),\gamma}}{\hat{F}_{(1),\gamma}^\alpha}\right),\ \hat{F}_{(\alpha),\gamma} = \left(\frac{\sum_{j=1}^k |y_j|^\gamma}{kG(\alpha,\gamma)}\right)^{\alpha/\gamma},\quad \hat{F}_{(1),\gamma} = \left(\frac{\sum_{j=1}^k |x_j|^\gamma}{kG(1,\gamma)}\right)^{1/\gamma}$$

which, after simplification, becomes

$$\hat{T}_{\alpha,\gamma} = \frac{1}{\alpha-1}\left(1 - \left(\frac{\sum_{j=1}^k |y_j|^\gamma}{\sum_{j=1}^k |x_j|^\gamma}\frac{G(1,\gamma)}{G(\alpha,\gamma)}\right)^{\alpha/\gamma}\right) \tag{17}$$

Recall $G(\alpha,\gamma)$ is defined in (8), and $\frac{G(1,0.5)}{G(\alpha,0.5)} = \frac{\sqrt{\pi}}{\Gamma\left(1-\frac{1}{2\alpha}\right)}$.

To better understand $\hat{F}_{(\alpha),\gamma}$, recall if $Z \sim S(\alpha,1)$, then $E|Z|^\gamma = G(\alpha,\gamma) < \infty$ if $-1 < \gamma < \alpha$. Therefore, $\left(\frac{\sum_{j=1}^k |y_j|^\gamma}{kG(\alpha,\gamma)}\right)$ is an unbiased estimate of $F_{(\alpha)}^{\gamma/\alpha}$. To recover $F_{(\alpha)}$, we need to apply the power $\alpha/\gamma$ operation. Thus, it is clear that, as long as $0 < \lambda < 1$, $\hat{F}_{(\alpha),\gamma}$ is a consistent estimator of $F_{(\alpha)}$ and $E\left(\hat{F}_{(\alpha),\gamma}\right)$ is finite. In particular, the variance of $\hat{F}_{(\alpha),\gamma}$ is bounded if $0 < \gamma < 0.5$:

$$E\left(\hat{F}_{(\alpha),\gamma}\right) = F_{(\alpha)} + O\left(\frac{1}{k}\right),\qquad Var\left(\hat{F}_{(\alpha),\gamma}\right) = \frac{F_{(\alpha)}^2}{k}\frac{\alpha^2}{\gamma^2}\frac{G(\alpha,2\gamma) - G^2(\alpha,\gamma)}{G^2(\alpha,\gamma)} + O\left(\frac{1}{k^2}\right)$$

The variance is unbounded if $\gamma = 0.5$ and $\alpha = 1$, because $G(1,1) = \infty$ ($\Gamma(0) = \infty$). Interestingly, when $\gamma \to 0$ and $\alpha = 1$, the asymptotic variance reaches the minimum. In fact, when $\gamma \to 0$, $\hat{F}_{(\alpha),\gamma}$ converges to the geometric mean estimator $\hat{F}_{(\alpha),gm}$. A variant of $\hat{F}_{(\alpha),\gamma}$ was discussed in [12].

## 4.1 Theoretical Analysis

Based on Lemma 3 and Lemma 4 (which is a fairly technical proof), we know that the variance of the general estimator is essentially $Var\left(\hat{T}_{\alpha,\gamma}\right) = O\left(\frac{\Delta^{2\gamma-1}}{k}\right)$, for fixed $\gamma \in (0, 1/2)$. In other words, when $\gamma$ is close to 0, the variance of the entropy estimator is essentially on the order of $O\left(1/(k\Delta)\right)$, and while $\gamma$ is close to 1/2, the variance is essentially $O(1/k)$ as desired.

**Lemma 3** *For any fixed $\gamma \in (0, 1)$,*

$$Var\left(\hat{T}_{\alpha,\gamma}\right) = \frac{1}{\Delta^2} \frac{O\left(E(|x_1|^\gamma - |y_1|^\gamma)^2\right)}{k} + O\left(\frac{1}{k^2}\right) \qquad \square$$

**Lemma 4** *Let $0 < \Delta \leq 1/2$ and $\alpha = 1 + \Delta$. Let $\gamma \in (0, 1/2)$ and $m$ be a positive integer no smaller than $1/\gamma$. Then, there exists a universal constant $M$ such that*

$$E\left(|x_1|^\gamma - |y_1|^\gamma\right)^2 \leq M F_{(1)}^{2\gamma} \Delta^{1+2\gamma-1/m} m^2 \left\{m + \widetilde{H}_{(2m)}^2 + (1-2\gamma)^{-2}\right\} \Big/ (1 - 2\gamma),$$

*where $\widetilde{H}_{2m} = \left(\sum_{i=1}^D \frac{|A_t[i]|}{F_{(1)}}\left(\log \frac{|A_t[i]|}{F_{(1)}}\right)^{2m}\right)^{1/(2m)}$.* $\qquad \square$

We should clarify that our theoretical analysis is only applicable for fixed $\gamma \in (0, 1/2)$. When $\gamma = 0.5$, the estimator $\hat{T}_{(\alpha),0.5}$ is still well-behaved, except we are unable to precisely analyze this case. Also, since we do not compute the exact constant, it is possible that for some carefully chosen $\alpha$ (data-dependent), $\hat{T}_{(\alpha),\gamma}$ with $\gamma < 0.5$ may exhibit smaller variance than $\hat{T}_{(\alpha),0.5}$. We recommend $\hat{T}_{(\alpha),0.5}$ for convenience because it essentially frees practitioners from carefully choosing $\alpha$.

## 4.2 Experimental Results

Figure 3 presents some empirical results, for testing the general estimator $\hat{T}_{\alpha,\gamma}$ (17), using more word vector pairs (including the same 2 pairs in Figure 2). We can see that when $\gamma = 0.5$, the (normalized) MSEs become flat (as desired) as $\Delta = \alpha - 1 \to 0$. When $\gamma > 1/2$, the MSEs increase although the curves remain flat. When $\gamma < 1/2$, the MSEs blow up with increasing $\Delta$. Note that, when $\gamma < 1/2$, it is possible to achieve smaller MSEs if we carefully choose $\alpha$.

How many samples ($k$) are needed? If the goal is to estimate the Shannon entropy within a few percentages of the the true value, then $k = 100 \sim 1000$ should be sufficient, because $\sqrt{MSE}/H < 0.1$ when $k \geq 100$ as shown in Figure 3.

## 5 Conclusion

Entropy estimation is an important task in machine learning, data mining, network measurement, anomaly detection, neural computations, etc. In modern applications, the data are often generated in a streaming fashion and many operations on the streams can only be conducted in one-pass of the data. It has been a challenging problem to estimate the Shannon entropy of data streams.

The prior work [21] achieved some success in entropy estimation using symmetric stable random projections. However, even after aggressively exploiting the bias-variance tradeoff, they still need to a large number of samples, e.g., $10^6$, which is prohibitive in both time and space, especially considering that in streaming applications the pseudo-random numbers have to be re-generated on the fly, the cost of which is directly proportional to the sample size.

In our approach, we approximate the Shannon entropy using two high correlated estimates of the frequent comments. The positive correlation can substantially reduce the variance of the Shannon entropy estimate. However, finding the appropriate estimator of the frequency moment is another challenging task. We successfully find such an estimator and show that its variance (of the Shannon entropy estimate) is very small. Experimental results demonstrate that about $100 \sim 1000$ samples should be sufficient for achieving high accuracies.

## Acknowledgement

The research of Ping Li is partially supported by NSF-IIS-1249316, NSF-DMS-0808864, NSF-SES-1131848, and ONR-YIP-N000140910911. The research of Cun-Hui Zhang is partially supported by NSF-DMS-0906420, NSF-DMS-1106753, NSF-DMS-1209014, and NSA-H98230-11-1-0205.

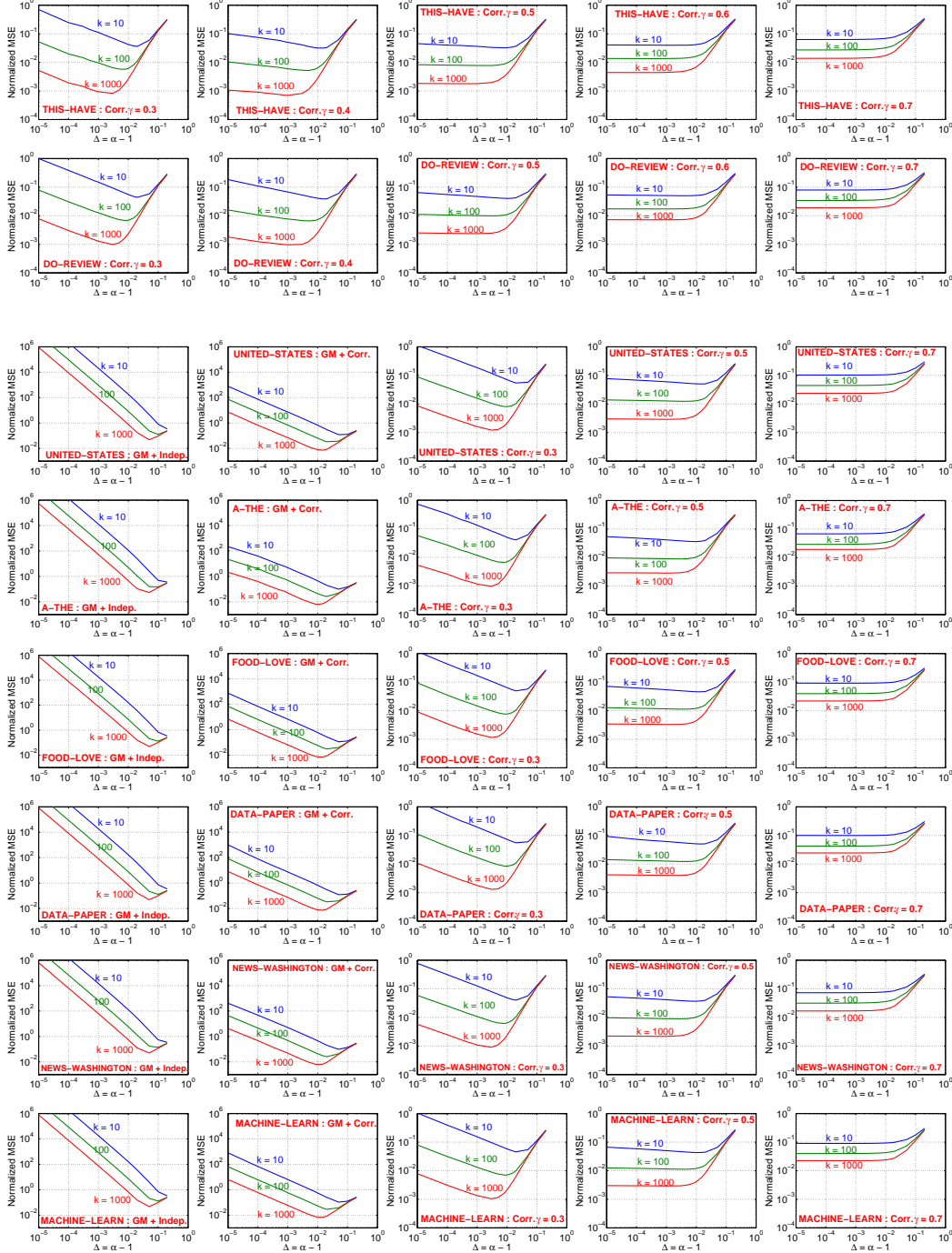

Figure 3: The **first two rows** are the normalized MSEs for same two vectors used in Figure 2, for estimating Shannon entropy using the general estimator $\hat{T}_{\alpha,\gamma}$ with $\gamma = 0.3, 0.4, 0.5, 0.6, 0.7$. For the **rest of the rows**, the leftmost panels are the results of using independent samples (i.e., the prior work [21]) and the geometric mean estimator. The second column of panels are the results of using correlated samples and the geometric mean estimator. The right three columns of panels are for the proposed general estimator $\hat{T}_{\alpha,\gamma}$ with $\gamma = 0.3, 0.5, 0.7$. We recommend $\gamma = 0.5$.

## Footnotes

[1][21] used $\frac{F_{(1+\Delta)} - F_{(1-\Delta)}}{2\Delta}$ and estimated the two frequency moments independently. The subtle difference between the finite difference approximations is not essential. It is the correlation that plays the crucial role.

# References

[1] Brian Babcock, Shivnath Babu, Mayur Datar, Rajeev Motwani, and Jennifer Widom. Models and issues in data stream systems. In *PODS*, pages 1–16, Madison, WI, 2002.

[2] Daniela Brauckhoff, Bernhard Tellenbach, Arno Wagner, Martin May, and Anukool Lakhina. Impact of packet sampling on anomaly detection metrics. In *IMC*, pages 159–164, Rio de Janeriro, Brazil, 2006.

[3] John M. Chambers, C. L. Mallows, and B. W. Stuck. A method for simulating stable random variables. *Journal of the American Statistical Association*, 71(354):340–344, 1976.

[4] Laura Feinstein, Dan Schnackenberg, Ravindra Balupari, and Darrell Kindred. Statistical approaches to DDoS attack detection and response. In *DARPA Information Survivability Conference and Exposition*, pages 303–314, 2003.

[5] Anupam Gupta, John D. Lafferty, Han Liu, Larry A. Wasserman, and Min Xu. Forest density estimation. In *COLT*, pages 394–406, Haifa, Israel, 2010.

[6] Nicholas J. A. Harvey, Jelani Nelson, and Krzysztof Onak. Streaming algorithms for estimating entropy. In *ITW*, 2008.

[7] Piotr Indyk. Stable distributions, pseudorandom generators, embeddings, and data stream computation. *Journal of ACM*, 53(3):307–323, 2006.

[8] Anukool Lakhina, Mark Crovella, and Christophe Diot. Mining anomalies using traffic feature distributions. In *SIGCOMM*, pages 217–228, Philadelphia, PA, 2005.

[9] Ashwin Lall, Vyas Sekar, Mitsunori Ogihara, Jun Xu, and Hui Zhang. Data streaming algorithms for estimating entropy of network traffic. In *SIGMETRICS*, pages 145–156, Saint Malo, France, 2006.

[10] Ping Li. Estimators and tail bounds for dimension reduction in $l_\alpha$ $(0 < \alpha \leq 2)$ using stable random projections. In *SODA*, pages 10 – 19, San Francisco, CA, 2008.

[11] Ping Li. Compressed counting. In *SODA*, New York, NY, 2009.

[12] Ping Li and Trevor J. Hastie. A unified near-optimal estimator for dimension reduction in $l_\alpha$ $(0 < \alpha \leq 2)$ using stable random projections. In *NIPS*, Vancouver, BC, Canada, 2007.

[13] Ping Li and Cun-Hui Zhang. A new algorithm for compressed counting with applications in shannon entropy estimation in dynamic data. In *COLT*, 2011.

[14] Qiaozhu Mei and Kenneth Church. Entropy of search logs: How hard is search? with personalization? with backoff? In *WSDM*, pages 45 – 54, Palo Alto, CA, 2008.

[15] S. Muthukrishnan. Data streams: Algorithms and applications. *Foundations and Trends in Theoretical Computer Science*, 1:117–236, 2 2005.

[16] Noam Nisan. Pseudorandom generators for space-bounded computations. In *Proceedings of the twenty-second annual ACM symposium on Theory of computing*, STOC, pages 204–212, 1990.

[17] Liam Paninski. Estimation of entropy and mutual information. *Neural Comput.*, 15(6):1191–1253, 2003.

[18] Constantino Tsallis. Possible generalization of boltzmann-gibbs statistics. *Journal of Statistical Physics*, 52:479–487, 1988.

[19] Kuai Xu, Zhi-Li Zhang, and Supratik Bhattacharyya. Profiling internet backbone traffic: behavior models and applications. In *SIGCOMM '05: Proceedings of the 2005 conference on Applications, technologies, architectures, and protocols for computer communications*, pages 169–180, Philadelphia, Pennsylvania, USA, 2005.

[20] Qiang Yang and Xindong Wu. 10 challeng problems in data mining research. *International Journal of Information Technology and Decision Making*, 5(4):597–604, 2006.

[21] Haiquan Zhao, Ashwin Lall, Mitsunori Ogihara, Oliver Spatscheck, Jia Wang, and Jun Xu. A data streaming algorithm for estimating entropies of od flows. In *IMC*, San Diego, CA, 2007.

[22] Vladimir M. Zolotarev. *One-dimensional Stable Distributions*. American Mathematical Society, Providence, RI, 1986.

